# KLD-Sampling: Adaptive Particle Filters

**Dieter Fox**
Department of Computer Science & Engineering
University of Washington
Seattle, WA 98195
Email: fox@cs.washington.edu

## Abstract

Over the last years, particle filters have been applied with great success to a variety of state estimation problems. We present a statistical approach to increasing the efficiency of particle filters by adapting the size of sample sets on-the-fly. The key idea of the KLD-sampling method is to bound the approximation error introduced by the sample-based representation of the particle filter. The name KLD-sampling is due to the fact that we measure the approximation error by the Kullback-Leibler distance. Our adaptation approach chooses a small number of samples if the density is focused on a small part of the state space, and it chooses a large number of samples if the state uncertainty is high. Both the implementation and computation overhead of this approach are small. Extensive experiments using mobile robot localization as a test application show that our approach yields drastic improvements over particle filters with fixed sample set sizes and over a previously introduced adaptation technique.

## 1 Introduction

Estimating the state of a dynamic system based on noisy sensor measurements is extremely important in areas as different as speech recognition, target tracking, mobile robot navigation, and computer vision. Over the last years, particle filters have been applied with great success to a variety of state estimation problems (see [3] for a recent overview). Particle filters estimate the posterior probability density over the state space of a dynamic system [4, 11]. The key idea of this technique is to represent probability densities by sets of samples. It is due to this representation, that particle filters combine efficiency with the ability to represent a wide range of probability densities. The efficiency of particle filters lies in the way they place computational resources. By sampling in proportion to likelihood, particle filters focus the computational resources on regions with high likelihood, where things really matter.

So far, however, an important source for increasing the efficiency of particle filters has only rarely been studied: *Adapting the number of samples over time*. While variable sample sizes have been discussed in the context of genetic algorithms [10] and interacting particle filters [2], most existing approaches to particle filters use a fixed number of samples during the whole state estimation process. This can be highly inefficient, since the complexity of the probability densities can vary drastically over time. An adaptive approach for particle filters has been applied by [8] and [5]. This approach adjusts the number of samples based on the likelihood of observations, which has some important shortcomings, as we will show.

In this paper we introduce a novel approach to adapting the number of samples over time. Our technique determines the number of samples based on statistical bounds on the sample-based approximation quality. Extensive experiments using a mobile robot indicate that our approach yields significant improvements over particle filters with fixed sample set sizes and over a previously introduced adaptation technique. The remainder of this paper is organized as follows: In the next section we will outline the basics of particle filters and their application to mobile robot localization. In Section 3, we will introduce our novel technique to adaptive particle filters. Experimental results are presented in Section 4 before we conclude in Section 5.

## 2 Particle filters for Bayesian filtering and robot localization

Particle filters address the problem of estimating the state $x$ of a dynamical system from sensor measurements. The goal of particle filters is to estimate a posterior probability density over the state space conditioned on the data collected so far. The data typically consists of an alternating sequence of time indexed observations $z_t$ and control measurements $u_t$, which describe the dynamics of the system. Let the belief $Bel(x_t)$ denote the posterior at time $t$. Under the Markov assumption, the posterior can be computed efficiently by recursively updating the belief whenever new information is received. Particle filters represent this belief by a set $S_t$ of $n$ weighted samples distributed according to $Bel(x_t)$:

$$S_t = \{\langle x_t^{(i)}, w_t^{(i)} \rangle \mid i = 1, \ldots, n\}$$

Here each $x_t^{(i)}$ is a sample (or state), and the $w_t^{(i)}$ are non-negative numerical factors called *importance weights*, which sum up to one. The basic form of the particle filter updates the belief according to the following sampling procedure, often referred to as sequential importance sampling with re-sampling (SISR, see also [4, 3]):

**Re-sampling:** Draw with replacement a random sample $x_{t-1}^{(i)}$ from the sample set $S_{t-1}$ according to the (discrete) distribution defined through the importance weights $w_{t-1}^{(i)}$. This sample can be seen as an instance of the belief $Bel(x_{t-1})$.

**Sampling:** Use $x_{t-1}^{(i)}$ and the control information $u_{t-1}$ to sample $x_t^{(j)}$ from the distribution $p(x_t \mid x_{t-1}, u_{t-1})$, which describes the dynamics of the system. $x_t^{(j)}$ now represents the density given by the product $p(x_t \mid x_{t-1}, u_{t-1}) Bel(x_{t-1})$. This density is the *proposal distribution* used in the next step.

**Importance sampling:** Weight the sample $x_t^{(j)}$ by the importance weight $p(z_t \mid x_t^{(j)})$, the likelihood of the sample $x_t^{(j)}$ given the measurement $z_t$.

Each iteration of these three steps generates a sample drawn from the posterior belief $Bel(x_t)$. After $n$ iterations, the importance weights of the samples are normalized so that they sum up to 1. It can be shown that this procedure in fact approximates the posterior density, using a sample-based representation [4, 2, 3].

### Particle filters for mobile robot localization

We use the problem of mobile robot localization to illustrate and test our approach to adaptive particle filters. Robot localization is the problem of estimating a robot's pose relative to a map of its environment. This problem has been recognized as one of the most fundamental problems in mobile robotics [1]. The mobile robot localization problem comes in different flavors. The simplest localization problem is position tracking. Here the initial robot pose is known, and localization seeks to correct small, incremental errors in a robot's odometry. More challenging is the global localization problem, where a robot is not told its initial pose, but instead has to determine it from scratch.

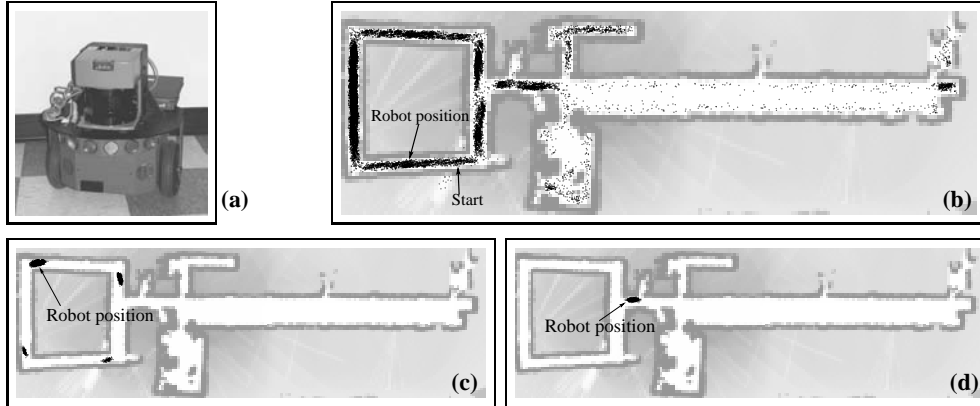

Fig. 1: a) Pioneer robot used throughout the experiments. b)-d) Map of an office environment along with a series of sample sets representing the robot's belief during global localization using sonar sensors (samples are projected into 2D). The size of the environment is 54m × 18m. b) After moving 5m, the robot is still highly uncertain about its position and the samples are spread trough major parts of the free-space. c) Even as the robot reaches the upper left corner of the map, its belief is still concentrated around four possible locations. d) Finally, after moving approximately 55m, the ambiguity is resolved and the robot knows where it is. All computation can be carried out in real-time on a low-end PC.

In the context of robot localization, the state $x_t$ of the system is the robot's position, which is typically represented in a two-dimensional Cartesian space and the robot's heading direction. The state transition probability $p(x_t \mid x_{t-1}, u_{t-1})$ describes how the position of the robot changes using information $u_t$ collected by the robot's wheel encoders. The perceptual model $p(y_t \mid x_t)$ describes the likelihood of making the observation $y_t$ given that the robot is at location $x_t$. In most applications, measurements consist of range measurements or camera images (see [6] for details). Figure 1 illustrates particle filters for mobile robot localization. Shown there is a map of a hallway environment along with a sequence of sample sets during global localization. In this example, all sample sets contain 100,000 samples. While such a high number of samples might be needed to accurately represent the belief during early stages of localization (cf. 1(a)), it is obvious that only a small fraction of this number suffices to track the position of the robot once it knows where it is (cf. 1(c)). Unfortunately, it is not straightforward how the number of samples can be adapted on-the-fly, and this problem has only rarely been addressed so far.

## 3  Adaptive particle filters with variable sample set sizes

The localization example in the previous section illustrates that the efficiency of particle filters can be greatly increased by changing the number of samples over time. Before we introduce our approach to adaptive particle filters, let us first discuss an existing technique.

### 3.1  Likelihood-based adaptation

We call this approach likelihood-based adaptation since it determines the number of samples such that the sum of non-normalized likelihoods (importance weights) exceeds a pre-specified threshold. This approach has been applied to dynamic Bayesian networks [8] and mobile robot localization [5]. The intuition behind this approach can be illustrated in the robot localization context: If the sample set is well in tune with the sensor reading, each individual importance weight is large and the sample set remains small. This is typically the case during position tracking (cf. 1(c)). If, however, the sensor reading carries a lot of surprise, as is the case when the robot is globally uncertain or when it lost track of its position, the

individual sample weights are small and the sample set becomes large.

The likelihood-based adaptation directly relates to the property that the variance of the importance sampler is a function of the mismatch between the proposal distribution and the distribution that is being approximated. Unfortunately, this mismatch is not always an accurate indicator for the necessary number of samples. Consider, for example, the ambiguous belief state consisting of four distinctive sample clusters shown in Fig. 1(b). Due to the symmetry of the environment, the average likelihood of a sensor measurement observed in this situation is approximately the same as if the robot knew its position unambiguously (cf. 1(c)). Likelihood-based adaptation would therefore use the same number of samples in both situations. Nevertheless, it is obvious that an accurate approximation of the belief shown in Fig. 1(b) requires a multiple of the samples needed to represent the belief in Fig. 1(c).

## 3.2 KLD-sampling

The key idea of our approach is to bound the error introduced by the sample-based representation of the particle filter. To derive this bound, we assume that the true posterior is given by a discrete, piecewise constant distribution such as a discrete density tree or a multi-dimensional histogram [8, 9]. For such a representation we can determine the number of samples so that the distance between the maximum likelihood estimate (MLE) based on the samples and the true posterior does not exceed a pre-specified threshold $\varepsilon$. We denote the resulting approach the KLD-sampling algorithm since the distance between the MLE and the true distribution is measured by the Kullback-Leibler distance. In what follows, we will first derive the equation for determining the number of samples needed to approximate a discrete probability distribution (see also [12, 7]). Then we will show how to modify the basic particle filter algorithm so that it realizes our adaptation approach.

To see, suppose that $n$ samples are drawn from a discrete distribution with $k$ different bins. Let the vector $\underline{X} = (X_1, \ldots, X_k)$ denote the number of samples drawn from each bin. $\underline{X}$ is distributed according to a multinomial distribution, i.e. $\underline{X} =\sim \text{Multinomial}_k(n, \underline{p})$, where $\underline{p} = p_1 \ldots p_k$ specifies the probability of each bin. The maximum likelihood estimate of $\underline{p}$ is given by $\widehat{\underline{p}} = n^{-1}\underline{X}$. Furthermore, the likelihood ratio statistic $\lambda_n$ for testing $\underline{p}$ is

$$\log \lambda_n = \sum_{j=1}^{k} X_j \log \left(\frac{\widehat{p}_j}{p_j}\right) = n \sum_{j=1}^{k} \widehat{p}_j \log \left(\frac{\widehat{p}_j}{p_j}\right). \tag{1}$$

When $\underline{p}$ is the true distribution, the likelihood ratio converges to a chi-square distribution:

$$2\log \lambda_n \to_d \chi^2_{k-1} \qquad \text{as} \qquad n \to \infty \tag{2}$$

Please note that the sum in the rightmost term of (1) specifies the K-L distance $K(\widehat{\underline{p}}, \underline{p})$ between the MLE and the true distribution. Now we can determine the probability that this distance is smaller than $\varepsilon$, given that $n$ samples are drawn from the true distribution:

$$P_{\underline{p}}(K(\widehat{\underline{p}}, \underline{p}) \leq \epsilon) \quad = \quad P_{\underline{p}}(2nK(\widehat{\underline{p}}, \underline{p}) \leq 2n\epsilon) \quad \doteq \quad P(\chi^2_{k-1} \leq 2n\epsilon) \tag{3}$$

The second step in (3) follows by replacing $nK(\widehat{\underline{p}}, \underline{p})$ with the likelihood ratio statistic, and by the convergence result in (2). The quantiles of the chi-square distribution are given by

$$P(\chi^2_{k-1} \leq \chi^2_{k-1,1-\delta}) = 1 - \delta. \tag{4}$$

Now if we choose $n$ such that $2n\epsilon$ is equal to $\chi^2_{k-1,1-\delta}$, we can combine (3) and (4) to get

$$P_{\underline{p}}(K(\widehat{\underline{p}}, \underline{p}) \leq \epsilon) \quad \doteq \quad 1 - \delta. \tag{5}$$

This derivation can be summarized as follows: If we choose the number of samples $n$ as

$$n = \frac{1}{2\epsilon}\chi^2_{k-1,1-\delta}, \tag{6}$$

then we can guarantee that with probability $1 - \delta$, the K-L distance between the MLE and the true distribution is less than $\varepsilon$. In order to determine $n$ according to (6), we need to compute the quantiles of the chi-square distribution. A good approximation is given by the Wilson-Hilferty transformation [7], which yields

$$ n \; = \; \frac{1}{2\epsilon}\chi^2_{k-1,1-\delta} \; \doteq \; \frac{k-1}{2\epsilon} \left\{ 1 - \frac{2}{9(k-1)} + \sqrt{\frac{2}{9(k-1)}}z_{1-\delta} \right\}^3 , \tag{7} $$

where $z_{1-\delta}$ is the upper $1 - \delta$ quantile of the standard normal $N(0,1)$ distribution.

This concludes the derivation of the sample size needed to approximate a discrete distribution with an upper bound $\varepsilon$ on the K-L distance. From (7) we see that the required number of samples is proportional to the inverse of the $\varepsilon$ bound, and to the first order linear in the number $k$ of bins with support. Here we assume that a bin of the multinomial distribution has support if its probability is above a certain threshold. This way the number $k$ will decrease with the certainty of the state estimation [1].

It remains to be shown how to apply this result to particle filters. The problem is that we do not know the true posterior distribution (the estimation of this posterior is the main goal of the particle filter). Fortunately, (7) shows that we do not need the complete discrete distribution but that it suffices to determine the number $k$ of bins with support. However, we do not know this quantity before we actually generate the distribution. Our approach is to estimate $k$ by counting the number of bins with support *during sampling*. To be more specific, we estimate $k$ for the proposal distribution $p(x_t \mid x_{t-1}, u_{t-1})Bel(x_{t-1})$ resulting from the first two steps of the particle filter update. The determination of $k$ can be done efficiently by checking for each generated sample whether it falls into an empty bin or not. Sampling is stopped as soon as the number of samples exceeds the threshold specified in (7). An update step of the resulting KLD-sampling particle filter is given in Table 1.

The implementation of this modified particle filter is straightforward. The only difference to the original algorithm is that we have to keep track of the number $k$ of supported bins. The bins can be implemented either as a fixed, multi-dimensional grid, or more efficiently as tree structures [8, 9]. Please note that the sampling process is guaranteed to terminate, since for a given bin size $\Delta$, the maximum number $k$ of bins is limited.

## 4 Experimental results

We evaluated our approach using data collected with one of our robots (see Figure 1). The data consists of a sequence of sonar scans and odometry measurements annotated with time-stamps to allow systematic real-time evaluations. In all experiments we compared our KLD-sampling approach to the likelihood-based approach discussed in Section 3.1, and to particle filters with fixed sample set sizes. Throughout the experiments we used different parameters for the three approaches. For the fixed approach we varied the number of samples, for the likelihood-based approach we varied the threshold used to determine the number of samples, and for our approach we varied $\varepsilon$, the bound on the K-L distance. In all experiments, we used a value of 0.99 for $\delta$ and a fixed bin size $\Delta$ of 50cm $\times$ 50cm $\times$ 10deg. We limited the maximum number of samples for all approaches to 100,000.

**Inputs:** $S_{t-1} = \{\langle x_{t-1}^{(i)}, w_{t-1}^{(i)}\rangle \mid i = 1, \ldots, n\}$ representing belief $Bel(x_{t-1})$,
control measurement $u_{t-1}$, observation $z_t$, bounds $\varepsilon$ and $\delta$, bin size $\Delta$

$S_t := \emptyset,\ n = 0,\ k = 0,\ \alpha = 0$        /* Initialize */

**do**        /* Generate samples ... */

    Sample an index $j(n)$ from the discrete distribution given by the weights in $S_{t-1}$
    Sample $x_t^{(n)}$ from $p(x_t \mid x_{t-1}, u_{t-1})$ using $x_{t-1}^{(j(n))}$ and $u_{t-1}$

    $w_t^{(n)} := p(z_t \mid x_t^{(n)});$        /* Compute importance weight */
    $\alpha := \alpha + w_t^{(n)}$        /* Update normalization factor */
    $S_t := S_t \cup \{\langle x_t^{(n)}, w_t^{(n)}\rangle\}$        /* Insert sample into sample set */

    **if** $(x_t^{(n)}$ falls into empty bin $b)$ **then**     /* Update number of bins with support */
        $k := k + 1$
        $b := $ non-empty

    $n := n + 1$        /* Update number of generated samples */
**while** $(n < \frac{1}{2\epsilon}\chi^2_{k-1, 1-\delta})$        /* ... until K-L bound is reached */

**for** $i := 1, \ldots, n$ **do**        /* Normalize importance weights */
    $w_t^{(i)} := w_t^{(i)}/\alpha$

**return** $S_t$

Table 1: KLD-sampling algorithm.

## Approximation of the true posterior

In the first set of experiments we evaluated how accurately the different methods approximate the true posterior density. Since the ground truth for these posteriors is not available, we compared the sample sets generated by the different approaches with reference sample sets. These reference sets were generated using a particle filter with a fixed number of 200,000 samples (far more than actually needed for position estimation). After each iteration, we computed the K-L distance between the sample sets and the corresponding reference sets, using histograms for both sets. Note that in these experiments the time-stamps were ignored and the algorithms was given as much time as needed to process the data. Fig. 2(a) plots the average K-L distance along with 95% confidence intervals against the average number of samples for the different algorithms (for clarity, we omitted the large error bars for K-L distances above 1.0). Each data point represents the average of 16 global localization runs with different start positions of the robot (each run itself consists of approximately 150 sample set comparisons at the different points in time). As expected, the more samples are used, the better the approximation. The curves also illustrate the superior performance of our approach: While the fixed approach requires about 50,000 samples before it converges to a K-L distance below 0.25, our approach converges to the same level using only 3,000 samples on average. This is also an improvement by a factor of 12 compared to the approximately 36,000 samples needed by the likelihood-based approach. In essence, these experiments indicate that our approach, even though based on several approximations, is able to accurately track the true posterior using significantly smaller sample sets on avarage than the other approaches.

## Real-time performance

Due to the computational overhead for determining the number of samples, it is not clear that our approach yields better results under real-time conditions. To test the performance of our approach under realistic conditions, we performed multiple global localization experiments under real-time considerations using the timestamps in the data sets. Again, the

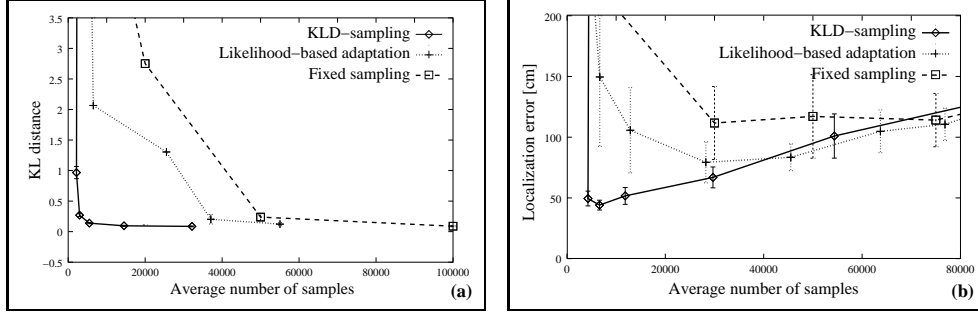

Fig. 2: The $x$-axis represents the average sample set size for different parameters of the three approaches. a) The $y$-axis plots the K-L distance between the reference densities and the sample sets generated by the different approaches (real-time constraints were not considered in this experiment). b) The $y$-axis represents the average localization error measured by the distance between estimated positions and reference positions. The U-shape in b) is due to the fact that under real-time conditions, an increasing number of samples results in higher update times and therefore loss of sensor data.

different average numbers of samples for KLD-sampling were obtained by varying the $\varepsilon$-bound. The minimum and maximum numbers of samples correspond to $\varepsilon$-bounds of 0.4 and 0.015, respectively. As a natural measure of the performance of the different algorithms, we determined the distance between the estimated robot position and the corresponding reference position after each iteration. [2] The results are shown in Fig. 2(b). The U-shape of all three graphs nicely illustrates the trade-off involved in choosing the number of samples under real-time constraints: Choosing not enough samples results in a poor approximation of the underlying posterior and the robot frequently fails to localize itself. On the other hand, if we choose too many samples, each update of the algorithm can take several seconds and valuable sensor data has to be discarded, which results in less accurate position estimates. Fig. 2(b) also shows that even under real-time conditions, our KLD-sampling approach yields drastic improvements over both fixed sampling and likelihood-based sampling. The smallest average localization error is 44cm in contrast to an average error of 79cm and 114cm for the likelihood-based and the fixed approach, respectively. This result is due to the fact that our approach is able to determine the best mix between more samples during early stages of localization and less samples during position tracking. Due to the smaller sample sets, our approach also needs significantly less processing power than any of the other approaches.

## 5   Conclusions and Future Research

We presented a statistical approach to adapting the sample set size of particle filters on-the-fly. The key idea of the KLD-sampling approach is to bound the error introduced by the sample-based belief representation of the particle filter. At each iteration, our approach generates samples until their number is large enough to guarantee that the K-L distance between the maximum likelihood estimate and the underlying posterior does not exceed a pre-specified bound. Thereby, our approach chooses a small number of samples if the density is focused on a small subspace of the state space, and chooses a large number of samples if the samples have to cover a major part of the state space.

Both the implementational and computational overhead of this approach are small. Extensive experiments using mobile robot localization as a test application show that our approach yields drastic improvements over particle filters with fixed sample sets and over a previously introduced adaptation approach [8, 5]. In our experiments, KLD-sampling yields bet-

ter approximations using only 6% of the samples required by the fixed approach, and using less than 9% of the samples required by the likelihood adaptation approach. So far, KLD-sampling has been tested using robot localization only. We conjecture, however, that many other applications of particle filters can benefit from this method.

KLD-sampling opens several directions for future research. In our current implementation we use a discrete distribution with a *fixed* bin size to determine the number of samples. We assume that the performance of the filter can be further improved by changing the discretization over time, using coarse discretizations when the uncertainty is high, and fine discretizations when the uncertainty is low. Our approach can also be extended to the case where in certain parts of the state space, highly accurate estimates are needed, while in other parts a rather crude approximation is sufficient. This problem can be addressed by locally adapting the discretization to the desired approximation quality using multi-resolution tree structures [8, 9] in combination with stratified sampling. As a result, more samples are used in "important" parts of the state space, while less samples are used in other parts. Another area of future research is the thorough investigation of particle filters under real-time conditions. In many applications the rate of incoming sensor data is higher than the update rate of the particle filter. This introduces a trade-off between the number of samples and the amount of sensor data that can be processed (cf. 2(b)). In our future work, we intend to address this problem using techniques similar to the ones introduced in this work.

### Acknowledgments

The author wishes to thank Jon A. Wellner and Vladimir Koltchinskii for their help in deriving the statistical background of this work. Additional thanks go to Wolfram Burgard and Sebastian Thrun for their valuable feedback on early versions of the technique.

## Footnotes

[1]This need for a threshold to determine $k$ (and to make $k$ vary over time) is not particularly elegant. However, it results in an efficient implementation that does not even depend on the value of the threshold itself (see next paragraph). We also implemented a version of the algorithm using the complexity of the state space to determine the number of samples. Complexity is measured by $2^H$, where $H$ is the entropy of the distribution. This approach does not depend on thresholding at all, but it does not have a guarantee of approximation bounds and does not yield significantly different results.

[2]Position estimates are extracted using histograming and local averaging, and the reference positions were determined by evaluating the robot's highly accurate laser range-finder information.

### References

[1] I. J. Cox and G. T. Wilfong, editors. *Autonomous Robot Vehicles*. Springer Verlag, 1990.

[2] P. Del Moral and L. Miclo. Branching and interacting particle systems approximations of feynamkac formulae with applications to non linear filtering. In *Seminaire de Probabilites XXXIV*, number 1729 in Lecture Notes in Mathematics. Springer-Verlag, 2000.

[3] A. Doucet, N. de Freitas, and N. Gordon, editors. *Sequential Monte Carlo in Practice*. Springer-Verlag, New York, 2001.

[4] A. Doucet, S.J. Godsill, and C. Andrieu. On sequential monte carlo sampling methods for Bayesian filtering. *Statistics and Computing*, 10(3), 2000.

[5] D. Fox, W. Burgard, F. Dellaert, and S. Thrun. Monte Carlo Localization: Efficient position estimation for mobile robots. In *Proc. of the National Conference on Artificial Intelligence (AAAI)*, 1999.

[6] D. Fox, S. Thrun, F. Dellaert, and W. Burgard. Particle filters for mobile robot localization. In Doucet et al. [3].

[7] N. Johnson, S. Kotz, and N. Balakrishnan. *Continuous univariate distributions*, volume 1. John Wiley & Sons, New York, 1994.

[8] D. Koller and R. Fratkina. Using learning for approximation in stochastic processes. In *Proc. of the International Conference on Machine Learning (ICML)*, 1998.

[9] A. W. Moore, J. Schneider, and K. Deng. Efficient locally weighted polynomial regression predictions. In *Proc. of the International Conference on Machine Learning (ICML)*, 1997.

[10] M. Pelikan, D.E. Goldberg, and E. Cant-Paz. Bayesian optimization algorithm, population size, and time to convergence. In *Proc. of the Genetic and Evolutionary Computation Conference (GECCO)*, 2000.

[11] M. K. Pitt and N. Shephard. Filtering via simulation: auxiliary particle filters. *Journal of the American Statistical Association*, 94(446), 1999.

[12] J.A. Rice. *Mathematical Statistics and Data Analysis*. Duxbury Press, second edition, 1995.
